# The Infinite Partially Observable Markov Decision Process

**Finale Doshi-Velez**
Cambridge University
Cambridge, CB21PZ, UK
`finale@alum.mit.edu`

## Abstract

The Partially Observable Markov Decision Process (POMDP) framework has proven useful in planning domains where agents must balance actions that provide knowledge and actions that provide reward. Unfortunately, most POMDPs are complex structures with a large number of parameters. In many real-world problems, both the structure and the parameters are difficult to specify from domain knowledge alone. Recent work in Bayesian reinforcement learning has made headway in learning POMDP models; however, this work has largely focused on learning the parameters of the POMDP model. We define an infinite POMDP (iPOMDP) model that does not require knowledge of the size of the state space; instead, it assumes that the number of visited states will grow as the agent explores its world and only models visited states explicitly. We demonstrate the iPOMDP on several standard problems.

## 1 Introduction

The Partially Observable Markov Decision Process (POMDP) model has proven attractive in domains where agents must reason in the face of uncertainty because it provides a framework for agents to compare the values of actions that gather information and actions that provide immediate reward. Unfortunately, modelling real-world problems as POMDPs typically requires a domain expert to specify both the structure of the problem and a large number of associated parameters, and both of which are often difficult tasks. Current methods in reinforcement learning (RL) focus on learning the parameters online, that is, while the agent is acting in its environment. Bayesian RL [1, 2, 3] has recently received attention because it allows the agent to reason both about uncertainty in its model of the environment and uncertainty within environment itself. However, these methods also tend to focus on learning parameters of an environment rather than the structure.

In the context of POMDP learning, several algorithms [4, 5, 6, 7] have applied Bayesian methods to reason about the unknown model parameters. All of these approaches provide the agent with the size of the underlying state space and focus on learning the transition and observation[1] dynamics for each state. Even when the size of the state space is known, however, just making the agent reason about a large number of unknown parameters at the beginning of the learning process is fraught with difficulties. The agent has insufficient experience to fit a large number of parameters, and therefore much of the model will be highly uncertain. Trying to plan under vast model uncertainty often requires significant computational resources; moreover, the computations are often wasted effort when the agent has very little data. Using a point estimate of the model instead—that is, ignoring the model uncertainty—can be highly inaccurate if the expert's prior assumptions are a poor match for the true model.

We propose a nonparametric approach to modelling the structure of the underlying space—specifically, the number of states in the agent's world—which allows the agent to start with a simple model and grow it with experience. Building on the infinite hidden Markov model (iHMM) [8], the infinite POMDP (iPOMDP) model posits that the environment contains of an unbounded number of states. The agent is expected to stay in a local region; however, as time passes, it may explore states that it has not visited before. Initially, the agent will infer simple, local models of the environment corresponding to its limited experience (also conducive to fast planning). It will dynamically add structure as it accumulates evidence for more complex models. Finally, a data-driven approach to structure discovery allows the agent to agglomerate states with identical dynamics (see section 4 for a toy example).

## 2 The Infinite POMDP Model

A POMDP consists of the n-tuple $\{S,A,O,T,\Omega,R,\gamma\}$. $S$, $A$, and $O$ are sets of states, actions, and observations. The transition function $T(s'|s,a)$ defines the distribution over next-states $s'$ to which the agent may transition after taking action $a$ from state $s$. The observation function $\Omega(o|s',a)$ is a distribution over observations $o$ that may occur in state $s'$ after taking action $a$. The reward function $R(s,a)$ specifies the immediate reward for each state-action pair (see figure 1 for a slice of the graphical model). The factor $\gamma \in [0,1)$ weighs the importance of current and future rewards.

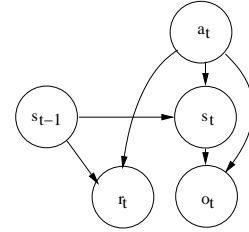

We focus on discrete state and observation spaces (generalising to continuous observations is straightforward) and finite action spaces. The size of the state space is unknown and potentially unbounded. The transitions, observations, and rewards are modelled with an iHMM.

Figure 1: A time-slice of the POMDP model.

**The Infinite Hidden Markov Model** A standard hidden Markov model (HMM) consists of the n-tuple $\{S,O,T,\Omega\}$, where the transition $T(s'|s)$ and observation $\Omega(o|s')$ distributions only depend on the hidden state. When the number of hidden states is finite and discrete, Dirichlet distributions may be used as priors over the transition and observation distributions. The iHMM [9] uses a hierarchical Dirichlet Process (HDP) to define a prior over HMMs where the number of underlying states is unbounded.[2] To generate a model from the iHMM prior, we:

1. Draw the mean transition distribution $\bar{T} \sim \mathsf{Stick}(\lambda)$.
2. Draw observations $\Omega(\cdot|s,a) \sim H$ for each $s$, $a$.
3. Draw transitions $T(\cdot|s,a) \sim \mathsf{DP}(\alpha,\bar{T})$ for each $s$, $a$.

where $\lambda$ is the DP concentration parameter and $H$ is a prior over observation distributions. For example, if the observations are discrete, then $H$ could be a Dirichlet distribution.

Intuitively, the first two steps define the observation distribution and an overall popularity for each state. The second step uses these overall state popularities to define individual state transition distributions. More formally, the first two steps involve a draw $G_0 \sim \mathsf{DP}(\lambda,H)$, where the atoms of $G_0$ are $\Omega$, and $\bar{T}$ are the associated stick-lengths.[3] Recall that in the stick breaking procedure, the $s^{th}$ stick-length, $\bar{T}_s$, is given by $v_s \prod_{i=1}^{s-1}(1-v_i)$, where $v_i \sim \mathsf{Beta}(1,\lambda)$. While the number of states is unbounded, $\bar{T}_s$ decreases exponentially with $s$, meaning that "later" states are less popular. This construction of $\bar{T}_s$ also ensures that $\sum_s^\infty \bar{T}_s = 1$. The top part of figure 2 shows a cartoon of a few elements of $\bar{T}$ and $\Omega$.

The second step of the iHMM construction involves defining the transition distributions $T(\cdot|s) \sim \mathsf{DP}(\alpha,\bar{T})$ for each state $s$, where $\alpha$, the concentration parameter for the DP, determines how closely the sampled distribution $T(\cdot|s)$ matches the mean transition distribution $\bar{T}$. Because $\bar{T}$ puts higher probabilities on states with smaller indices, $T(s'|s)$ will also generally put more mass on earlier $s'$ (see lower rows of figure 2). Thus, the generating process encodes a notion that the agent will spend most of its time in some local region. However, the longer the agent acts in this infinite space, the more likely it is to transition to somewhere new.

**Infinite POMDPs** To extend the iHMM framework to iPOMDPs, we must incorporate actions and rewards into the generative model. To incorporate actions, we draw an observation distribution $\Omega(\cdot|s,a) \sim H$ for each action $a$ and each state $s$. Similarly, during the second step of the generative process, we draw a transition distribution $T(s'|s,a) \sim \text{DP}(\alpha, \bar{T})$ for each state-action pair.[4]

HMMs have one output—observations—while POMDPs also output rewards. We treat rewards as a secondary set of observations. For this work, we assume that the set of possible reward values is given, and we use a multinomial distribution to describe the probability $R(r|s,a)$ of observing reward $r$ after taking action $a$ in state $s$. As with the observations, the reward distributions $R$ are drawn from Dirichlet distribution $H_R$. We use multinomial distributions for convenience; however, other reward distributions (such as Gaussians) are easily incorporate in this framework.

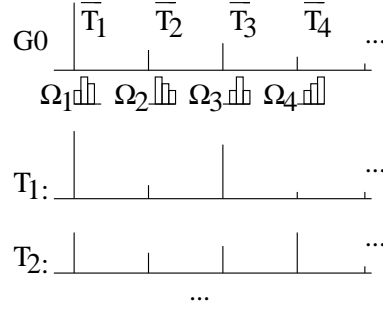

Figure 2: iHMM: The first row shows each state's observation distribution $\Omega_s$ and the mean transition distribution $\bar{T}$. Later rows show each state's transition distribution.

In summary, the iPOMDP prior requires that we specify

- a set of actions $A$ and observations $O$,
- a generating distribution $H$ for the observation distributions and $H_R$ for the rewards (these generating distributions can have any form; the choice will depend on the application),
- a mean transition concentration factor $\lambda$ and a state transition concentration factor $\alpha$, and
- a discount factor $\gamma$.

To sample a model from the iPOMDP prior, we first sample the mean transition distribution $\bar{T} \sim \text{Stick}(\lambda)$. Next, for each state $s$ and action $a$, we sample

- $T(\cdot|s,a) \sim \text{DP}(\alpha, \bar{T})$,
- $\Omega(\cdot|s,a) \sim H$,
- $R(\cdot|s,a) \sim H_R$.

Samples from the iPOMDP prior have an infinite number of states, but fortunately all of these states do not need to be explicitly represented. During a finite lifetime the agent can only visit a finite number of states, and thus the agent can only make inferences about a finite number of states. The remaining (infinite) states are equivalent from agent's perspective, as, in expectation, these states will exhibit the mean dynamics of the prior. Thus, the only parts of the infinite model that need to be initialised are those corresponding to the states the agent has visited as well as a catch-all state representing all other states. In reality, of course, the agent does not know the states it has visited: we discuss joint inference over the unknown state history and the model in section 3.1.

## 3 Planning

As in the standard Bayesian RL framework, we recast the problem of POMDP learning as planning in a larger 'model-uncertainty' POMDP in which both the true model and the true state are unknown. We outline below our procedure for planning in this joint space of POMDP models and unknown states and the detail each step—belief monitoring and action-selection—in sections 3.1 and 3.2.

Because the true state is hidden, the agent must choose its actions based only on past actions and observations. Normally the best action to take at time $t$ depends on the entire history of actions and observations that the agent has taken so far. However, the probability distribution over current states, known as the *belief*, is a sufficient statistic for a history of actions and observations. In discrete state spaces, the belief at time $t+1$ can be computed from the previous belief, $b_t$, the last action $a$, and observation $o$, by the following application of Bayes rule:

$$b_{t+1}^{a,o}(s) = \Omega(o|s,a) \sum_{s' \in S} T(s|s',a) b_t(s')/Pr(o|b,a), \tag{1}$$

where $Pr(o|b,a)=\sum_{s'\in S}\Omega(o|s',a)\sum_{s\in S}T(s'|s,a)b_t(s)$. However, it is intractable to express the joint belief $b$ over models and states with a closed-form expression. We approximate the belief $b$ with a set of sampled models $m = \{T, \Omega, R\}$, each with weight $w(m)$. Each model sample $m$ maintains a belief over states $b_m(s)$. The states are discrete, and thus the belief $b_m(s)$ can be updated using equation 1. Details for sampling the models $m$ are described in section 3.1.

Given the belief, the agent must choose what action to choose next. One approach is to solve the planning problem offline, that is, determine a good action for every possible belief. If the goal is to maximize the expected discounted reward, then the optimal policy is given by:

$$V_t(b) = \max_{a\in A}Q_t(b,a), \tag{2}$$

$$Q_t(b,a) = R(b,a) + \gamma\sum_{o\in O}Pr(o|b,a)V_t(b^{a,o}), \tag{3}$$

where the value function $V(b)$ is the expected discounted reward that an agent will receive if its current belief is $b$ and $Q(b,a)$ is the value of taking action $a$ in belief $b$. The exact solution to equation 3 is only tractable for tiny problems, but many approximation methods [12, 13, 14] have been developed to solve POMDPs offline.

While we might hope to solve equation 3 over the state space of a single model, it is intractable to solve over the joint space of states and infinite models—the model space is so large that standard point-based approximations will generally fail. Moreover, it makes little sense to find the optimal policy for all models when only a few models are likely. Therefore, instead of solving 3 offline, we build a forward-looking search tree at each time step (see [15] for a review of forward search in POMDPs). The tree computes the value of action by investigating a number of steps into the future. The details of the action selection are discussed in section 3.2.

## 3.1 Belief Monitoring

As outlined in section 3, we approximate the joint belief over states and models through a set of samples. In this section, we describe a procedure for sampling a set of models $m = \{T, \Omega, R\}$ from the true belief, or posterior, over models.[5] These samples can then be used to approximate various integrations over models that occur during planning; in the limit of infinite samples, the approximations will be guaranteed to converge to their true values. To simplify matters, we assume that given a model $m$, it is tractable to maintain a closed-form belief $b_m(s)$ over states using equation 1. Thus, models need to be sampled, but beliefs do not.

Suppose we have a set of models $m$ that have been drawn from the belief at time $t$. To get a set of models drawn from the belief at time $t+1$, we can either draw the models directly from the new belief or adjust the weights on the model set at time $t$ so that they now provide an accurate representation of the belief at time $t + 1$. Adjusting the weights is computationally most straightforward: directly following belief update equation 1, the importance weight $w(m)$ on model $m$ is given by:

$$w_{t+1}^{a,o}(m) \propto \Omega(o|m,a)w_t(m), \tag{4}$$

where $\Omega(o|m,a)=\sum_{s\in S}\Omega(o|s,m,a)b_m(s)$, and we have used $T(m'|m,a) = \delta_m(m')$ because the true model does not change.

The advantage of simply reweighting the samples is that the belief update is extremely fast. However, new experience may quickly render all of the current model samples unlikely. Therefore, we must periodically resample a new set of models directly from the current belief. The beam-sampling approach of [16] is an efficient method for drawing samples from an iHMM posterior. We adapt this approach to allow for observations with different temporal shifts (since the reward $r_t$ depends on the state $s_t$, whereas the observation $o_t$ is conditioned on the state $s_{t+1}$) and for transitions indexed by both the current state and the most recent action. The correctness of our sampler follows directly from the correctness of the beam sampler [16].

The beam-sampler is an auxiliary variable method that draws samples from the iPOMDP posterior. A detailed description of beam sampling is beyond the scope of this paper; however, we outline the general procedure below. The inference alternates between three phases:

- **Sampling slice variables to limit trajectories to a finite number of hidden states.** Given a transition model $T$ and a state trajectory $\{s_1, s_2, \ldots\}$, an auxiliary variable $u_t \sim \mathsf{Uniform}([0, \min(T(\cdot|s_t, a))])$ is sampled for each time $t$. The final column $k$ of the transition matrix is extended via additional stick-breaking until $\max(T(s_k|s, a)) < u_t.)$. Only transitions $T(s'|s, a) > u_t$ are considered for inference at time $t$.[6]
- **Sampling a hidden state trajectory.** Now that we have a finite model, we apply forward filtering-backward sampling (FFBS) [18] to sample the underlying state sequence.
- **Sampling a model.** Given a trajectory over hidden states, transition, observation, and reward distributions are sampled for the visited states (it only makes sense to sample distributions for visited states, as we do not have information about unvisited states). In this finite setting, we can resample the transitions $T(\cdot|s, a)$ using standard Dirichlet posteriors:

$$T(\cdot|s, a) \sim \mathsf{Dirichlet}(T_1^{sa} + n_1^{sa}, T_2^{sa} + n_2^{sa}, \ldots, T_k^{sa} + n_k^{sa}, \sum_{i=k+1}^{\infty} T_i^{sa}), \qquad (5)$$

where $k$ is the number of *active* or used states, $T_i^{sa}$ is the prior probability of transitioning to state $i$ from state $s$ after taking action $a$, and $n_i^{sa}$ is the number of observed transitions to state $i$ from $s$ after $a$. The observations and rewards are resampled in a similar manner: for example, if the observations are discrete with Dirichlet priors:

$$\Omega(\cdot|s, a) \sim \mathsf{Dirichlet}(H_1 + n^{o_1 sa}, H_2 + n^{o_2 sa}, \ldots, H_{|O|} + n^{o_{|O|} sa}). \qquad (6)$$

As with all MCMC methods, initial samples (from the *burn-in* period) are biased by sampler's start position; only after the sampler has mixed will the samples be representative of the true posterior.

Finally, we emphasize that the approach outline above is a sampling approach and not a maximum likelihood estimator; thus the samples, drawn from the agent's belief, capture the variation over possible models. The representation of the belief is necessarily approximate due to our use of samples, but the samples are drawn from the true current belief—no other approximations have been made. Specifically, we are not filtering: each run of the beam sampler produces samples from the current belief. Because they are drawn from the true posterior, all samples have equal weight.

### 3.2 Action Selection

Given a set of models, we apply a stochastic forward search in the model-space to choose an action. The general idea behind forward search [15] is to use a forward-looking tree to compute action-values. Starting from the agent's current belief, the tree branches on each action the agent might take and each observation the agent might see. At each action node, the agent computes its expected immediate reward $R(a) = \mathbb{E}_m[\mathbb{E}_{s|m}[R(\cdot|s, a)]]$.

From equation 3, the value of taking action $a$ in belief $b$ is

$$Q(a, b) = R(a, b) + \gamma \sum_o \Omega(o|b, a) \max_{a'} Q(a', b^{ao}) \qquad (7)$$

where $b^{ao}$ is the agent's belief after taking action $a$ and seeing observation $o$ from belief $b$. Because action selection must be completed online, we use equation 4 to update the belief over models via the weights $w(m)$. Equation 7 is evaluated recursively for each $Q(a', b^{ao})$ up to some depth $D$.

The number of evaluations grows with $(|A||O|)^D$, so doing a full expansion is feasible only for very small problems. We approximate the true value stochastically by sampling only a few observations from the distribution $P(o|a) = \sum_m P(o|a, m) w(m)$. Equation 7 reduces to

$$Q(a, b) = R(a, b) + \gamma \frac{1}{N_O} \sum_i \max_{a'} Q(a', b^{ao_i}) \qquad (8)$$

where $N_O$ is the number of sampled observations and $o_i$ is the $i^{th}$ sampled observation.

Once we reach a prespecified depth in the tree, we must approximate the value of the leaves. For each model $m$ in the leaves, we can compute the value $Q(a, b_m, m)$ of the action $a$ by approximately

solving offline the POMDP model that $m$ represents. We approximate the value of action $a$ as

$$Q(a, b) \approx \sum_m w(m) Q(a, b_m, m). \qquad (9)$$

This approximation is always an overestimate of the value, as it assumes that the uncertainty over models—but not the uncertainty over states—will be resolved in the following time step (similar to the QMDP [19] assumption).[7] As the iPOMDP posterior becomes peaked and the uncertainty over models decreases, the approximation becomes more exact.

The quality of the action selection largely follows from the bounds presented in [20] for planning through forward search. The key difference is that now our belief representation is particle-based; during the forward search we approximate an expected rewards over all possible models with rewards from the particles in our set. Because we can guarantee that our models are drawn from the true posterior over models, this approach is a standard Monte Carlo approximation of the expectation. Thus, we can apply the central limit theorem to state that the estimated expected rewards will be distributed around the true expectation with approximately normal noise $N(0, \frac{\sigma^2}{n})$, where $n$ is the number of POMDP samples and $\sigma^2$ is a problem-specific variance.

## 4  Experiments

We begin with a series of illustrative examples demonstrating the properties of the iPOMDP. In all experiments, the observations were given vague hyperparameters (1.0 Dirichlet counts per element), and rewards were given hyperparameters that encouraged peaked distributions (0.1 Dirichlet counts per element). The small counts on the reward hyperparameters encoded the prior belief that $R(\cdot|s, a)$ is highly peaked, that is, each state-action pair will likely have one associated reward value. Beliefs were approximated with sample set of 10 models. Models were resampled between episodes and reweighted during episodes. A burn-in of 500 iterations was used for the beam sampler when drawing these models directly from the belief. The forward-search was expanded to a depth of 3.

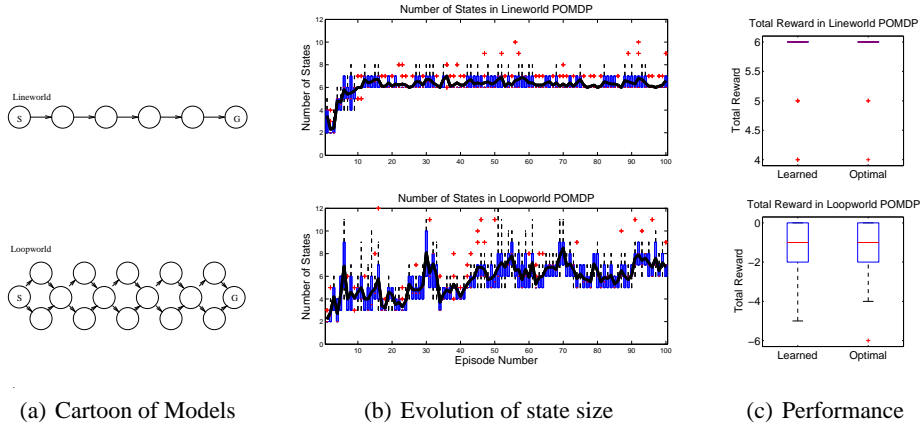

|  (a) Cartoon of Models  |  (b) Evolution of state size  |  (c) Performance  |

Figure 3: Various comparisons of the lineworld and loopworld models. Loopworld infers only necessary states, ignoring the more complex (but irrelevant) structure.

**Avoiding unnecessary structure: Lineworld and Loopworld.** We designed a pair of simple environments to show how the iPOMDP infers states only as it can distinguish them. The first, lineworld was a length-six corridor in which the agent could either travel left or right. Loopworld consisted of a corridor with a series of loops (see figure 3(a)); now the agent could travel though the upper or lower branches. In both environments, only the two ends of the corridors had unique observations.

Actions produced the desired effect with probability 0.95, observations were correct with probability 0.85 (that is, 15% of the time the agent saw an incorrect observation). The agent started at the left end of the corridor and received a reward of -1 until it reached the opposite end (reward 10).

The agent eventually infers that the lineworld environment consists of six states—based on the number of steps it requires to reach the goal—although in the early stages of learning it infers distinct states only for the ends of the corridor and groups the middle region as one state. The loopworld agent also shows a growth in the number of states over time (see figure 3(b)), but it never infers separate states for the identical upper and lower branches. By inferring states as they needed to explain its observations—instead of relying on a prespecified number of states—the agent avoided the need to consider irrelevant structure in the environment. Figure 3(c) shows that the agent (unsurprisingly) learns optimal performance in both environments.

**Adapting to new situations: Tiger-3.** The iPOMDP's flexibility also lets it adapt to new situations. In the tiger-3 domain, a variant of the tiger problem [19] the agent had to choose one of three doors to open. Two doors had tigers behind them ($r = -100$) and one door had a small reward ($r = 10$). At each time step, the agent could either open a door or listen for the "quiet" door. It heard the correct door correctly with probability 0.85.

The reward was unlikely to be behind the third door ($p = .2$), but during the first 100 episodes, we artificially ensured that the reward was always behind doors 1 or 2. The improving rewards in figure 4 show the agent steadily learning the dynamics of its world; it learned never to open door 3. The dip in 4 following episode 100 occurs when we next allowed the reward to be behind all three doors, but the agent quickly adapts to the new possible state of its environment. The iPOMDP enabled the agent to first adapt quickly to its simplified environment but add complexity when it was needed.

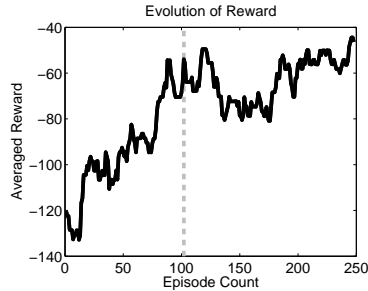

Figure 4: Evolution of reward from tiger-3.

**Broader Evaluation.** We next completed a set of experiments on POMDP problems from the literature. Tests had 200 episodes of learning, which interleaved acting and resampling models, and 100 episodes of testing with the models fixed. During learning, actions were chosen stochastically based on its value with probability 0.05 and completely randomly with probability 0.01. Otherwise, they were chosen greedily (we found this small amount of randomness was needed for exploration to overcome our very small sample set and search depths). We compared accrued rewards and running times for the iPOMDP agent against (1) an agent that knew the state count and used EM to train its model, (2) an agent that knew the state count and that used the same forward-filtering backward-sampling (FFBS) algorithm used in the beam sampling inner loop to sample models, and (3) an agent that used FFBS with ten times the true number of states. For situations where the number of states is not known, the last case is particularly interesting—we show that simply overestimating the number of states is not necessarily the most efficient solution.

Table 1 summarises the results. We see that the iPOMDP often infers a smaller number of states than the true count, ignoring distinctions that the history does not support. The middle three columns show the speeds of the three controls relative the iPOMDP. Because the iPOMDP generally uses smaller state spaces, we see that most of these values are greater than 1, indicating the iPOMDP is faster. (In the largest problem, dialog, the oversized FFBS model did not complete running in several days.) The latter four columns show accumulated rewards; we see that the iPOMDP is generally on par or better than the methods that have access to the true state space size. Finally, figure 5 plots the learning curve for one of problems, shuttle.

# 5  Discussion

Recent work in learning POMDP models include[23], which uses a set of Gaussian approximations to allow for analytic value function updates in the POMDP space, and [5], which jointly reasons over the space Dirichlet parameter and states when planning in discrete POMDPs. Sampling based approaches include Medusa [4], which learns using state-queries, and [7], which learns using policy

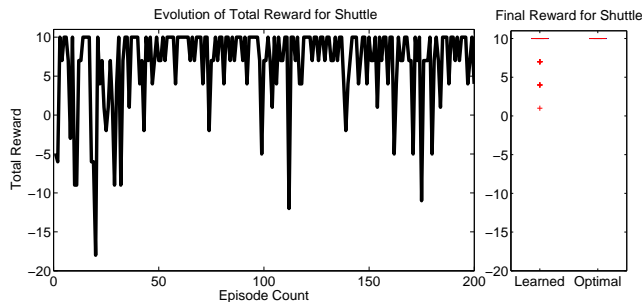

Figure 5: Evolution of reward for shuttle. During training (left), we see that the agent makes fewer mistakes toward the end of the period. The boxplots on the right show rewards for 100 trials after learning has stopped; we see the iPOMDP-agent's reward distribution over these 100 trials is almost identical an agent who had access to the correct model.

Table 1: Inferred states and performance for various problems. The iPOMDP agent (FFBS-Inf) often performs nearly as well as the agents who had knowledge of the true number of states (EM-true, FFBS-true), learning the necessary number of states much faster than an agent for which we overestimate the number of states (FFBS-big).

| Metric | States | | Relative Training Time | | | Performance | | | |
|---|---|---|---|---|---|---|---|---|---|
| Problem | True | FFBS-Inf | EM-true | FFBS-true | FFBS-big | EM-true | FFBS-true | FFBS-big | FFBS-Inf |
| Tiger[19] | 2 | 2.1 | 0.41 | 0.70 | 1.50 | -277 | 0.49 | 4.24 | 4.06 |
| Shuttle[21] | 8 | 2.1 | 1.82 | 1.02 | 3.56 | 10 | 10 | 10 | 10 |
| Network[19] | 7 | 4.36 | 1.56 | 1.09 | 4.82 | 1857 | 7267 | 6843 | 6508 |
| Gridworld[19] (adapted) | 26 | 7.36 | 3.57 | 2.48 | 59.1 | -25 | -51 | -67 | -13 |
| Dialog[22] (adapted) | 51 | 2 | 0.67 | 5.15 | - | -3023 | -1326 | - | -1009 |

queries. All of these approaches assume that the number of underlying states is known; all but [7] focus on learning only the transition and observation models.

In many problems, however, the underlying number of states may not be known—or may require significant prior knowledge to model—and, from the perspective of performance, is irrelevant. The iPOMDP model allows the agent to adaptively choose the complexity of the model; any expert knowledge is incorporated into the prior: for example, the Dirichlet counts on observation parameters can be used to give preference to certain observations as well as encode whether we expect observations to have low or high noise. As seen in the results, the iPOMDP allows the complexity of the model to scale gracefully with the agent's experience. Future work remains to tailor the planning to unbounded spaces and refine the inference for POMDP resampling.

Past work has attempted to take advantage of structure in POMDPs [24, 25], but learning that structure has remained an open problem. By giving the agent an unbounded state space—but strong locality priors—the iPOMDP provides one principled framework to learning POMDP structure. Moreover, the hierarchical Dirichlet process construction described in section 2 can be extended to include more structure and deeper hierarchies in the transitions.

# 6 Conclusion

We presented the infinite POMDP, a new model for Bayesian RL in partially observable domains. The iPOMDP provides a principled framework for an agent to posit more complex models of its world as it gains more experience. By linking the complexity of the model to the agent's experience, the agent is not forced to consider large uncertainties—which can be computationally prohibitive— near the beginning of the planning process, but it can later come up with accurate models of the world when it requires them. An interesting question may also to apply these methods to learning large MDP models within the Bayes-Adaptive MDP framework [26].

## Footnotes

[1] [7] also learns rewards.

[2]The iHMM models in [8] and [9] are formally equivalent [10].

[3]A detailed description of DPs and HDPs is beyond the scope of this paper; please refer to [11] for background on Dirichlet processes and [9] for an overview of HDPs.

[4]We use the same base measure $H$ to draw all observation distributions; however, a separate measures $H_a$ could be used for each action if one had prior knowledge about the expected observation distribution for reach action. Likewise, one could also draw a separate $\bar{T}_a$ for each action.

[5]We will use the words *posterior* and *belief* interchangeably; both refer to the probability distribution over the hidden state given some initial belief (or *prior*) and the history of actions and observations.

[6]For an introduction to slice sampling, refer to [17].

[7]We also experimented with approximating $Q(a, b) \approx 80 - \texttt{percentile}(\{w(m)Q(a, b_{m, m})\})$. Taking a higher percentile ranking as the approximate value places a higher value on actions with larger uncertainty. As the values of the actions become more well known and the discrepancies between the models decreases, this criterion reduces to the true value of the action.

# References

[1] R. Dearden, N. Friedman, and D. Andre, "Model based Bayesian exploration," pp. 150–159, 1999.

[2] M. Strens, "A Bayesian framework for reinforcement learning," in *ICML*, 2000.

[3] P. Poupart, N. Vlassis, J. Hoey, and K. Regan, "An analytic solution to discrete Bayesian reinforcement learning," in *ICML*, (New York, NY, USA), pp. 697–704, ACM Press, 2006.

[4] R. Jaulmes, J. Pineau, and D. Precup, "Learning in non-stationary partially observable Markov decision processes," ECML Workshop, 2005.

[5] S. Ross, B. Chaib-draa, and J. Pineau, "Bayes-adaptive POMDPs," in *Neural Information Processing Systems (NIPS)*, 2008.

[6] S. Ross, B. Chaib-draa, and J. Pineau, "Bayesian reinforcement learning in continuous POMDPs with application to robot navigation," in *ICRA*, 2008.

[7] F. Doshi, J. Pineau, and N. Roy, "Reinforcement learning with limited reinforcement: Using Bayes risk for active learning in POMDPs," in *International Conference on Machine Learning*, vol. 25, 2008.

[8] M. J. Beal, Z. Ghahramani, and C. E. Rasmussen, "The infinite hidden Markov model," in *Machine Learning*, pp. 29–245, MIT Press, 2002.

[9] Y. W. Teh, M. I. Jordan, M. J. Beal, and D. M. Blei, "Hierarchical Dirichlet processes," *Journal of the American Statistical Association*, vol. 101, no. 476, pp. 1566–1581, 2006.

[10] J. V. Gael and Z. Ghahramani, *Inference and Learning in Dynamic Models*, ch. Nonparametric Hidden Markov Models. Cambridge University Press, 2010.

[11] Y. W. Teh, "Dirichlet processes." Submitted to Encyclopedia of Machine Learning, 2007.

[12] J. Pineau, G. Gordon, and S. Thrun, "Point-based value iteration: An anytime algorithm for POMDPs," *IJCAI*, 2003.

[13] M. T. J. Spaan and N. Vlassis, "Perseus: Randomized point-based value iteration for POMDPs," *Journal of Artificial Intelligence Research*, vol. 24, pp. 195–220, 2005.

[14] T. Smith and R. Simmons, "Heuristic search value iteration for POMDPs," in *Proc. of UAI 2004*, (Banff, Alberta), 2004.

[15] S. Ross, J. Pineau, S. Paquet, and B. Chaib-Draa, "Online planning algorithms for POMDPs," *Journal of Artificial Intelligence Research*, vol. 32, pp. 663–704, July 2008.

[16] J. van Gael, Y. Saatci, Y. W. Teh, and Z. Ghahramani, "Beam sampling for the infinite hidden Markov model," in *ICML*, vol. 25, 2008.

[17] R. Neal, "Slice sampling," *Annals of Statistics*, vol. 31, pp. 705–767, 2000.

[18] C. K. Carter and R. Kohn, "On Gibbs sampling for state space models," *Biometrika*, vol. 81, pp. 541–553, September 1994.

[19] M. L. Littman, A. R. Cassandra, and L. P. Kaelbling, "Learning policies for partially observable environments: scaling up," *ICML*, 1995.

[20] D. McAllester and S. Singh, "Approximate planning for factored POMDPs using belief state simplification," in *UAI 15*, 1999.

[21] L. Chrisman, "Reinforcement learning with perceptual aliasing: The perceptual distinctions approach," in *In Proceedings of the Tenth National Conference on Artificial Intelligence*, pp. 183–188, AAAI Press, 1992.

[22] F. Doshi and N. Roy, "Efficient model learning for dialog management," in *Proceedings of Human-Robot Interaction (HRI 2007)*, (Washington, DC), March 2007.

[23] P. Poupart and N. Vlassis, "Model-based Bayesian reinforcement learning in partially observable domains," in *ISAIM*, 2008.

[24] J. H. Robert, R. St-aubin, A. Hu, and C. Boutilier, "SPUDD: Stochastic planning using decision diagrams," in *UAI*, pp. 279–288, 1999.

[25] A. P. Wolfe, "POMDP homomorphisms," in *NIPS RL Workshop*, 2006.

[26] M. O. Duff, *Optimal learning: computational procedures for Bayes-adaptive markov decision processes*. PhD thesis, 2002.

